# Reinforcement Learning Using Approximate Belief States

**Andrés Rodríguez** *
Artificial Intelligence Center
SRI International
333 Ravenswood Avenue, Menlo Park, CA 94025
*rodriguez@ai.sri.com*

**Ronald Parr, Daphne Koller**
Computer Science Department
Stanford University
Stanford, CA 94305
*{parr,koller}@cs.stanford.edu*

## Abstract

The problem of developing good policies for partially observable Markov decision problems (POMDPs) remains one of the most challenging areas of research in stochastic planning. One line of research in this area involves the use of reinforcement learning with belief states, probability distributions over the underlying model states. This is a promising method for small problems, but its application is limited by the intractability of computing or representing a full belief state for large problems. Recent work shows that, in many settings, we can maintain an *approximate belief state*, which is fairly close to the true belief state. In particular, great success has been shown with approximate belief states that marginalize out correlations between state variables. In this paper, we investigate two methods of full belief state reinforcement learning and one novel method for reinforcement learning using factored approximate belief states. We compare the performance of these algorithms on several well-known problem from the literature. Our results demonstrate the importance of approximate belief state representations for large problems.

## 1 Introduction

The Markov Decision Processes (MDP) framework [2] is a good way of mathematically formalizing a large class of sequential decision problems involving an agent that is interacting with an environment. Generally, an MDP is defined in such a way that the agent has complete knowledge of the underlying state of the environment. While this formulation poses very challenging research problems, it is still a very optimistic modeling assumption that is rarely realized in the real world. Most of the time, an agent must face uncertainty or incompleteness in the information available to it. An extension of this formalism that generalizes MDPs to deal with this uncertainty is given by partially observable Markov Decision Processes (POMDPs) [1, 11] which are the focus of this paper.

Solving a POMDP means finding an optimal behavior policy $\pi^*$, that maps from the agent's available knowledge of the environment, its *belief state*, to actions. This is usually done through a function, $V$, that assigns values to belief states. In the fully observable (MDP)

case, a value function can be computed efficiently for reasonably sized domains. The situation is somewhat different for POMDPs, where finding the optimal policy is PSPACE-hard in the number of underlying states [6]. To date, the best known exact algorithms to solve POMDPs are taxed by problems with a few dozen states [5].

There are several general approaches to approximating POMDP value functions using reinforcement learning methods and space does not permit a full review of them. The approach upon which we focus is the use of a belief state as a probability distribution over underlying model states. This is in contrast to methods that manipulate augmented state descriptions with finite memory [9, 12] and methods that work directly with observations [8].

The main advantage of a probability distribution is that it summarizes all of the information necessary to make optimal decisions [1]. The main disadvantages are that a model is required to compute a belief state, and that the task of representing and updating belief states in large problems is itself very difficult. In this paper, we do not address the problem of obtaining a model; our focus is on the the most effective way of using a model. Even with a known model, reinforcement learning techniques can be quite competitive with exact methods for solving POMDPs [10]. Hence, we focus on extending the model-based reinforcement learning approach to larger problems through the use of *approximate belief states*. There are risks to such an approach: inaccuracies introduced by belief state approximation could give an agent a hopelessly inaccurate perception of its relationship to the environment.

Recent work [4], however, presents an approximate tracking approach, and provides theoretical guarantees that the result of this process cannot stray too far from the exact belief state. In this approach, rather than maintaining an exact belief state, which is infeasible in most realistically large problems, we maintain an approximate belief state, usually from some restricted class of distributions. As the approximate belief state is updated (due to actions and observations), it is continuously projected back down into this restricted class. Specifically, we use *decomposed* belief states, where certain correlations between state variables are ignored.

In this paper we present empirical results comparing three approaches to belief state reinforcement learning. The most direct approach is the use of a neural network with one input for each element of the full belief state. The second is the SPOVA method [10], which uses a function approximator designed for POMDPs and the third is the use of a neural network with an approximate belief state as input. We present results for several well-known problems in the POMDP literature, demonstrating that while belief state approximation is ill-suited for some problems, it is an effective means of attacking large problems.

## 2   Basic Framework and Algorithms

A POMDP is defined as a tuple $< S, A, O, \mathcal{T}, \mathcal{R}, \mathcal{O} >$ of three sets and three functions. $S$ is a set of *states*, $A$ is a set of *actions* and $O$ is a set of *observations*. The *transition* function $\mathcal{T} : S \times A \rightarrow \Pi(S)$ specifies how the actions affect the state of the world. It can be viewed as $\mathcal{T}(s_i, a, s_j) = P(s_j|a, s_i)$, the probability that the agent reaches state $s_j$ if it currently is in state $s_i$ and takes action $a$. The *reward* function $\mathcal{R} : S \times A \rightarrow \mathbb{R}$ determines the immediate reward received by the agent The *observation* model $\mathcal{O} : S \times A \rightarrow \Pi(O)$ determines what the agent perceives, depending on the environment state and the action taken. $\mathcal{O}(s, a, o) = P(o|a, s)$ is the probability that the agent observes $o$ when it is in state $s$, having taken the action $a$.

## 2.1  POMDP belief states

A *belief state*, $b$, is defined as a probability distribution over all states $s \in S$, where $b(s)$, represents probability that the environment is in state $s$. After taking action $a$ and observing $o$, the belief state is updated using Bayes rule:

$$b'(s') = P(s' \mid a, o, b) = \frac{\mathcal{O}(s', a, o) \sum_{s_i \in S} \mathcal{T}(s_i, a, s')b(s_i)}{\sum_{s_j \in S} \mathcal{O}(s_j, a, o) \sum_{s_i \in S} \mathcal{T}(s_i, a, s_j)b(s_i)}$$

The size of an exact belief state is equal to the number of states in the model. For large problems, maintaining and manipulating an exact belief state can be problematic even if the the transition model has a compact representation [4]. For example, suppose the state space is described via a set of random variables $\mathbf{X} = \{X_1, \dots, X_n\}$, where each $X_i$ takes on values in some finite domain $\mathrm{Val}(X_i)$, a particular $s$ defines a value $x_i \in \mathrm{Val}(X_i)$ for each variable $X_i$. The full belief state representation will be exponential in $n$. We use the approximation method analyzed by Boyen and Koller [4], where the variables are partitioned into a set of disjoint clusters $\mathbf{C}_1 \dots \mathbf{C}_k$ and belief functions, $b_1 \dots b_k$ are maintained over the variables in each cluster. At each time step, we compute the exact belief state, then compute the individual belief functions by marginalizing out inter-cluster correlations. For some assignment, $\mathbf{c}_i$, to variables in $\mathbf{C}_i$, we obtain $b_i(\mathbf{c}_i) = \sum_{\mathbf{y} \notin \mathbf{c}_1} P(\mathbf{c}_i, \mathbf{y})$. An approximation of the original, full belief state is then reconstructed as $b(s) = \prod_{i=1}^{k} b_i(\mathbf{c}_i)$.

By representing the belief state as a product of marginal probabilities, we are projecting the belief state into a reduced space. While a full belief state representation for $n$ state variables would be exponential in $n$, the size of decomposed belief state representation is exponential in the size of the largest cluster and additive in the number of clusters. For processes that mix rapidly enough, the errors introduced by approximation will stay bounded over time [4]. As discussed by Boyen and Koller [4], this type of decomposed belief state is particularly suitable for processes that can themselves be factored and represented as a *dynamic Bayesian network* [3]. In such cases we can avoid ever representing an exponentially sized belief state. However, the approach is fully general, and can be applied in any setting where the state is defined as an assignment of values to some set of state variables.

## 2.2  Value functions and policies for POMDPs

If one thinks of a POMDP as an MDP defined over belief states, then the well-known fixed point equations for MDPs still hold. Specifically,

$$V^*(b) = \max_a \left[ \sum_{s \in S} b(s) \mathcal{R}(s, a) + \gamma \sum_{o \in O} P(o|a, b) V^*(b') \right]$$

where $\gamma$ is the discount factor and $b'$ (defined above) is the next belief state. The optimal policy is determined by the maximizing action for each belief state. In principle, we could use Q-learning or value iteration directly to solve POMDPs. The main difficulty lies in the fact that there are uncountably many belief states, making a tabular representation of the value function impossible.

Exact methods for POMDPs use the fact that finite horizon value functions are piecewise-linear and convex [11], ensuring a finite representation. While finite, this representation can grow exponentially with the horizon, making exact approaches impractical in most settings. Function approximation is an attractive alternative to exact methods. We implement function approximation using a set of parameterized Q-functions, where $Q_a(b, \mathbf{W}_a)$ is the reward-to-go for taking action $a$ in belief state $b$. A value function is reconstructed from the Q-functions as $V(b) = \max_a(Q_a(b, \mathbf{W}_a))$, and the update rule for $\mathbf{W}_a$ when a transition

from state $b$ to $b'$ under action $a$ with reward $R$ is:

$$\Delta \mathbf{W}_a = \alpha(\gamma V(b') + r - Q_a(b, \mathbf{W}_a))\nabla_{\mathbf{W}_a} Q_a(b, \mathbf{W}_a)$$

### 2.3 Function approximation architectures

We consider two types of function approximators. The first is a two-layer feedforward neural network with sigmoidal internal units and a linear outermost layer. We used one network for each Q function. For full belief state reinforcement learning, we used networks with $|\mathcal{S}|$ inputs (one for each component of the belief state) and $\sqrt{|\mathcal{S}|}$ hidden nodes. For approximate belief state reinforcement learning, we used networks with one input for each assignment to the variables in each cluster. If we had two clusters, for example, each with 3 binary variables, then our Q networks would each have $2^3 + 2^3 = 16$ inputs. We kept the number of hidden nodes for each network as the square root of the number of inputs.

Our second function approximator is SPOVA [10], which is a soft max function designed to exploit the piecewise-linear structure of POMDP value functions. A SPOVA Q function maintains a set of weight vectors $\mathbf{w}_{a1} \ldots \mathbf{w}_{ai}$, and is evaluated as:

$$Q_a(b) = \sqrt[k]{\sum_i (b \cdot \mathbf{W}_{a_i})^k}$$

In practice, a small value of $k$ (usually 1.2) is adopted at the start of learning, making the function very smooth. This is increased during learning until SPOVA closely approximates a PWLC function of $b$ (usually $k = 8$). We maintained one SPOVA Q function for each action and assigned $\sqrt{|\mathcal{S}|}$ vectors to each function. This gave $O(|\mathcal{A}||\mathcal{S}|\sqrt{|\mathcal{S}|})$ parameters to both SPOVA and the full belief state neural network.

## 3 Empirical Results

We present results on several problems from the POMDP literature and present an extension to a known machine repair problem that is designed to highlight the effects of approximate belief states. Our results are presented in the form of performance graphs, where the value of the current policy is obtained by taking a snapshot of the value function and measuring the discounted sum of reward obtained by the resulting policy in simulation. We use "NN" to refer to the neural network trained reinforcement learner trained with the full belief state and the term "Decomposed NN" to refer to the neural network trained with an approximate belief which is decomposed as a product of marginals. We used a simple exploration strategy, starting with a 0.1 probability of acting randomly, which decreased linearly to 0.01.

Due to space limitations, we are not able to describe each model in detail. However, we used publicly available model description files from [5].[1] Table 3.4 shows the running times of the different methods. These are generally much lower than what would be required to solve these problems using exact methods.

### 3.1 Grid Worlds

We begin by considering two grid worlds, a $4 \times 3$ world from [10] and a 60-state world from [7]. The $4 \times 3$ world contains only 11 states and does not have a natural decomposition into state variables, so we compared SPOVA only with the full belief state neural network.

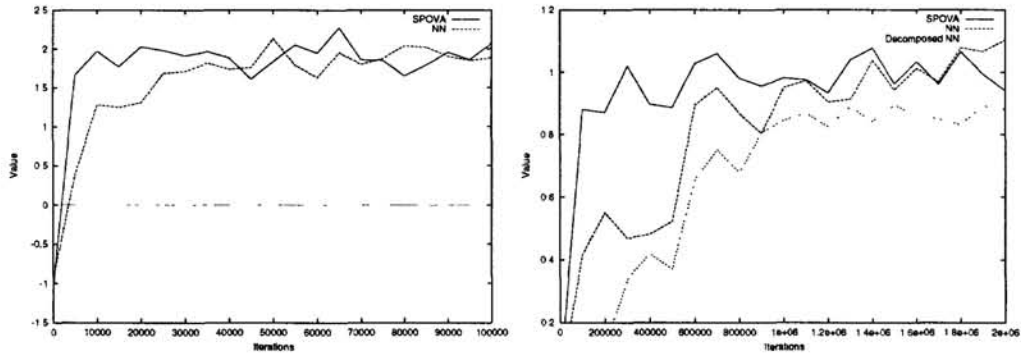

Figure 1: a) 3 × 4 Grid World, b) 60-state maze

The experimental results, which are averaged over 25 training runs and 100 simulations per policy snapshot, are presented in Figure 1a. They show that SPOVA learns faster than the neural network, but that the network does eventually catch up.

The 60-state robot navigation problem [7] was amenable to a decomposed belief state approximation since its underlying state space comes from the product of 15 robot positions and 4 robot orientations. We decomposed the belief state with two clusters, one containing a position state variable and the other containing an orientation state variable. Figure 1b shows results in which SPOVA again dominates. The decomposed NN has trouble with this problem because the effects of position and orientation on the value function are not easily decoupled, i.e., the effect of orientation on value is highly state-dependent. This meant that the decomposed NN was forced to learn a much more complicated function of its inputs than the function learned by the network using the full belief state.

## 3.2 Aircraft Identification

Aircraft identification is another problem studied in Cassandra's thesis. It includes sensing actions for identifying incoming aircraft and actions for attacking threatening aircraft. Attacks against friendly aircraft are penalized, as are failures to intercept hostile aircraft. This is a challenging problem because there is tension in deciding between the various sensors. Better sensors tend to make the base more visible to hostile aircraft, while more stealthy sensors are less accurate. The sensors give information about both the aircraft's type and distance from the base.

The state space of this problem is comprised of three main components. `aircraft type` — either the aircraft is a `friend` or it is a `foe`; `distance` — how far the aircraft is currently from the base discretized into an adjustable number, $d$, of distinct distances; `visibility` — a measure of how visible the base is to the approaching aircraft, which is discretized into 5 levels.

We chose $d = 10$, gaving this problem 104 states. The problem has a natural decomposition into state variables for aircraft type, distance and base visibility. The results for the three algorithms are shown in Figure 2(a). This is the first problem where we start to see an advantage from decomposing the belief state. For the decomposed NN, we used three separate clusters, one for each variable, which meant that the network had only 17 inputs. Not only did the simpler network learn faster, but it learned a better policy overall. We believe that this illustrates an important point: even though SPOVA and the full belief state neural network may be more expressive than the decomposed NN, the decomposed NN is able to search the space of functions it can represent much more efficiently due to the reduced number of parameters.

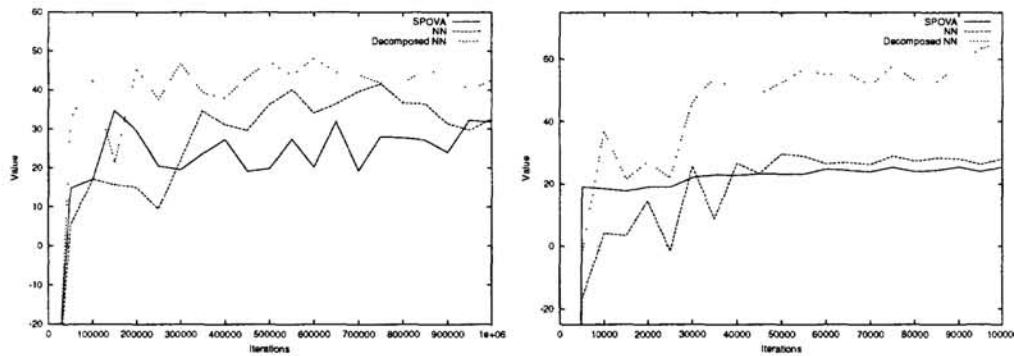

Figure 2: a) Aircraft Identification, b) Machine Maintenance

### 3.3 Machine Maintenance

Our last problem was the machine maintenance problem from Cassandra's database. The problem assumes that there is a machine with a certain number of components. The quality of the parts produced by the machine is determined by the condition of the components. Each component can be in one of four conditions: good — the component is in good condition; fair — the component has some amount of wear, and would benefit from some maintenance; bad — the part is very worn and could use repairs; broken — the part is broken and must be replaced. The status of the components is observable only if the machine is completely disassembled.

Figure 2(b) shows performance results for this problem for the 4 component version of this problem. At 256 states, it was at the maximum size for which a full belief state approach was manageable. However, the belief state for this problem decomposes naturally into clusters describing the status of each machine, creating a decomposed belief state with just four components. The graph shows the dominance of this this simple decomposition approach. We believe that this problem clearly demonstrates the advantage of belief state decomposition: The decomposed NN learns a function of 16 inputs in fraction of the time it takes for the full net or SPOVA to learn a lower-quality function of 256 inputs.

### 3.4 Running Times

The table below shows the running times for the different problems presented above. These are generally much less than what would be required to solve these problems exactly. The full NN and SPOVA are roughly comparable, but the decomposed neural network is considerably faster. We did not exploit any problem structure in our approximate belief state computation, so the time spent computing belief states is actually *larger* for the decomposed NN. The savings comes from the the reduction in the number of parameters used, which reduced the number of partial derivatives computed. We expect the savings to be significantly more substantial for processes represented in a factored way [3], as the approximate belief state propagation algorithm can also take advantage of this additional structure.

## 4   Concluding Remarks

We have a proposed a new approach to belief state reinforcement learning through the use of approximate belief states. Using well-known examples from the POMDP literature, we have compared approximate belief state reinforcement learning with two other methods

| Problem | SPOVA | NN | Decomposed NN |
|---|---|---|---|
| 3x4 | 19.1 s | 13.0 s | |
| Hallway | 32.8 min | 47.1 min | 3.2 min |
| Aircraft ID | 38.3 min | 49.9 min | 4.4 min |
| Machine M. | 2.5 h | 2.6 h | 4.7 min |

Table 1: Run times (in seconds, minutes or hours) for the different algorithms

that use exact belief states. Our results demonstrate that, while approximate belief states may not be ideal for tightly coupled problem features, such as the position and orientation of a robot, they are a natural and effective means of addressing some large problems. Even for the medium-sized problems we showed here, approximate belief state reinforcement learning can outperform full belief state reinforcement learning using fewer trials and much less CPU time. For many problems, exact belief state methods will simply be impractical and approximate belief states will provide a tractable alternative.

## Acknowledgements

This work was supported by the ARO under the MURI program "Integrated Approach to Intelligent Systems," by ONR contract N66001-97-C-8554 under DARPA's HPKB program, and by the generosity of the Powell Foundation and the Sloan Foundation.

## Footnotes

*The work presented in this paper was done while the first author was at Stanford University.

[1]See http://www.cs.brown.edu/research/ai/pomdp/index.html. Note that this file format specifies a starting distribution for each problem and our results are reported with respect to this starting distribution.

## References

[1] K. J. Astrom. Optimal control of Markov decision processes with incomplete state estimation. *J. Math. Anal. Applic.*, 10:174–205, 1965.

[2] R.E. Bellman. *Dynamic Programming*. Princeton University Press, 1957.

[3] C. Boutilier, T. Dean, and S. Hanks. Decision theoretic planning: Structural assumptions and computational leverage. *Journal of Artificial Intelligence Research*, 1999.

[4] X. Boyen and D. Koller. Tractable inference for complex stochastic processes. In *Proc. UAI*, 1998.

[5] A. Cassandra. *Exact and approximate Algorithms for partially observable Markov Decision Problems*. PhD thesis, Computer Science Dept., Brown Univ., 1998.

[6] M. Littman. *Algorithms for Sequential Decision Making*. PhD thesis, Computer Science Dept., Brown Univ., 1996.

[7] M. Littman, A. Cassandra, and L.P. Kaelbling. Learning policies for partially observable environments: Scaling up. In *Proc. ICML*, pages 362–370, 1996.

[8] J. Loch and S. Singh. Using eligibility traces to find the best memoryless policy in partially observable markov decision processes. In *Proc. ICML*. Morgan Kaufmann, 1998.

[9] Andrew R. McCallum. Overcoming incomplete perception with utile distinction memory. In *Proc. ICML*, pages 190–196, 1993.

[10] Ronald Parr and Stuart Russell. Approximating optimal policies for partially observable stochastic domains. In *Proc. IJCAI*, 1995.

[11] R. D. Smallwood and E. J. Sondik. The optimal control of partially observable Markov processes over a finite horizon. *Operations Research*, 21:1071–1088, 1973.

[12] M. Wiering and J. Schmidhuber. HQ-learning: Discovering Markovian subgoals for non-Markovian reinforcement learning. Technical report, Istituo Dalle Molle di Studi sull'Intelligenza Artificiale, 1996.